# Adaptive Passive-Aggressive Framework for Online Regression with Side Information

**Runhao Shi**, **Jiaxi Ying**, **Daniel P. Palomar**
The Hong Kong University of Science and Technology
`{rshiaf, jx.ying}@connect.ust.hk, palomar@ust.hk`

## Abstract

The Passive-Aggressive (PA) method is widely used in online regression problems for handling large-scale streaming data, typically updating model parameters in a passive-aggressive manner based on whether the error exceeds a predefined threshold. However, this approach struggles with determining optimal thresholds and adapting to complex scenarios with side information, where tracking accuracy is not the sole metric in the regression model. To address these challenges, we introduce a novel adaptive framework that allows finer adjustments to the weight vector in PA using side information. This framework adaptively selects the threshold parameter in PA, theoretically ensuring convergence to the optimal setting. Additionally, we present an efficient implementation of our algorithm that significantly reduces computational complexity. Numerical experiments show that our model achieves outstanding performance associated with the side information while maintaining low tracking error, demonstrating marked improvements over traditional PA methods across various scenarios.

## 1 Introduction

Online learning techniques, initially introduced by Zinkevich (2003), have gained significant popularity due to their robustness in adversarial environments and efficiency in processing large streaming data (Shalev-Shwartz et al., 2012; Orabona, 2019; Hazan, 2022). In the online learning framework, an online player continuously makes decisions and receives corresponding losses, aiming to minimize regret. Regret, in this context, refers to the worst-case discrepancy in performance compared to the best-fixed decision in hindsight, measuring the overhead of identifying the best-fixed decision.

These techniques have found widespread application in modeling regression problems for streaming data, enabling practical applications across various fields (Herbster, 2001; Crammer et al., 2006; Shalev-Shwartz and Ben-David, 2014). They are extensively applied in diverse domains such as portfolio selection (Li et al., 2012; Li and Hoi, 2012), malicious URL detection (Ma et al., 2009; Zhao and Hoi, 2013), and time series prediction (Anava et al., 2013, 2015; Hazan et al., 2018; Lale et al., 2020; Tsiamis and Pappas, 2022; Zhang et al., 2024). Without relying on strong assumptions, online regression models demonstrate robustness with regret guarantees in challenging scenarios. Furthermore, their incremental learning schemes make them highly adaptable to streaming data, eliminating the need to retrain the entire dataset and resulting in significant efficiency advantages.

One well-known online regression method is the Passive-Aggressive (PA) algorithm (Crammer et al., 2006). PA employs a passive-aggressive updating scheme to learn a weight vector for linear regression problems. It passively maintains the previous weight below a certain threshold and aggressively updates the weight when the loss exceeds the threshold. However, determining an appropriate threshold can be challenging. A small threshold prioritizes real-time tracking accuracy but may lead to overfitting and sensitivity to noise, compromising long-term tracking accuracy. Additionally, the selected weight may impact factors beyond accuracy in practical model performance. When

additional metrics and side information are available for evaluating performance, PA may struggle to achieve a more nuanced weight selection.

To address the aforementioned challenges, we propose an Adaptive Passive-Aggressive online regression framework with Side information (APAS) to achieve the following objectives:

- **Novel APAS framework:** We introduce a novel APAS framework that integrates side information into PA to enhance weight evaluation and selection. This framework adaptively selects the threshold parameter in PA, enabling it to achieve outstanding performance associated with the side information while maintaining a low tracking error.

- **Efficient algorithm:** We develop an efficient algorithm using the successive convex approximation (SCA, Scutari et al., 2013) to accelerate the computation of APAS. This algorithm rapidly converges to the optimal point, allowing flexibility in selecting measurements to integrate side information.

- **Regret bound:** We derive an $O(\sqrt{T})$ regret bound for our APAS framework for non-convex loss functions, ensuring the robustness and effectiveness of APAS theoretically. This regret bound matches the optimal regret bound for non-convex loss functions.

- **Extensive experiments:** We conduct an enhanced index tracking task on both synthetic and real financial datasets to validate the effectiveness and efficiency of APAS, which demonstrates the impressive performance of APAS in achieving high returns while maintaining small tracking errors.

**Notation:** Matrices and vectors are represented by bold letters. $[T]$ denotes the set $\{1, 2, \ldots, T\}$. The weight vector at time $t$ is denoted by $\mathbf{w}_t \in \mathcal{W}$. The instance and target in an online regression problem are denoted by $\mathbf{x}_t \in \mathbb{R}^N$ and $y_t \in \mathbb{R}$, respectively. The proximal operator and Moreau envelope associated with $\lambda h$ are denoted as $\text{prox}_{\lambda h}$ and $M_{\lambda h}$, respectively. The Euclidean projection of vector $\mathbf{u} \in \mathbb{R}^N$ onto the set $\mathcal{W}$ is denoted by $\Pi_{\mathcal{W}}(\mathbf{u}) = \arg\min_{\mathbf{w} \in \mathcal{W}} \|\mathbf{w} - \mathbf{u}\|_2^2$. For a continuous function $f(x)$, the set of subderivatives at point $a$ is denoted as $\partial f(a)$. The left derivative at $a$ is denoted by $\partial_- f(a) = \lim_{x \to a^-} \frac{f(x) - f(a)}{x - a}$. The derivative at point $a$, if it exists, is denoted as $f'(a)$.

## 2 Preliminaries

### 2.1 Online Learning

Online learning is a mathematical framework designed to address optimization problems where objective functions change over time. In this context, an online learner sequentially makes decisions $b_t$ based on historical loss and receives a new loss $f_t(b_t)$ after making the decision. The performance of an online learning algorithm is evaluated using the concept of regret $R_T$, which quantifies the discrepancy between the algorithm's performance and that of an optimal static parameter setting:

$$R_T = \sum_{t=1}^{T} f_t(b_t) - \min_{b \in \mathcal{X}} \left( \sum_{t=1}^{T} f_t(b) \right),$$

where $\mathcal{X}$ denotes the feasible set. An online learning strategy converges to the optimal static parameter setting if $R_T = o(T)$, indicating the average performance gap diminishes as the number of iterations $T$ approaches infinity. In the case of convex loss functions, different regularization functions can be employed to achieve various optimal regret bounds, depending on the assumptions about the curve of the loss function (Zinkevich, 2003; Hazan et al., 2007; Hazan and Seshadhri, 2007, 2009). Adaptive regularization methods, which select the regularization term dynamically, have also been proposed and widely adopted in various domains (Duchi et al., 2010, 2011; Van Erven and Koolen, 2016).

### 2.2 Passive-Aggressive Method

The Passive-Aggressive (PA) method is a popular online algorithm utilized for regression problems involving streaming data (Crammer et al., 2006). In an online regression problem, we receive an instance $\mathbf{x}_t \in \mathbb{R}^N$ and predict the target value $\mathbf{w}_t^\mathsf{T} \mathbf{x}_t$ using the incrementally learned vector $\mathbf{w}_t$, where the ground truth is $y_t$. PA predicts the next weight vector by solving the following optimization problem:

$$\widehat{\mathbf{w}}_{t+1} = \arg\min_{\mathbf{w} \in \mathbb{R}^N} \frac{1}{2} \|\mathbf{w} - \mathbf{w}_t\|_2^2 \qquad \text{subject to} \quad \ell_\varepsilon(\mathbf{w}; (\mathbf{x}_t, y_t)) = 0, \tag{1}$$

where $\ell_\varepsilon$ is the $\varepsilon$-insensitive hinge loss function defined as follows:

$$\ell_\varepsilon\left(\mathbf{w};(\mathbf{x},y)\right) = \begin{cases} 0 & \left|\mathbf{w}^{\mathsf{T}}\mathbf{x} - y\right| \le \varepsilon, \\ \left|\mathbf{w}^{\mathsf{T}}\mathbf{x} - y\right| - \varepsilon & \text{otherwise.} \end{cases}$$

Intuitively, PA performs an aggressive update when the discrepancy between the predicted value and the ground truth exceeds the threshold $\varepsilon$, and passively maintains the previous weight when the discrepancy is within the threshold $\varepsilon$. A smaller threshold may prioritize real-time tracking accuracy but could result in overfitting and compromise long-term performance. Therefore, the selection of the threshold $\varepsilon$ significantly influences the performance. Additionally, relying solely on tracking accuracy without considering side information may limit the method's potential performance.

## 3 Proposed Method

In this section, we present a novel framework that incorporates the side information into PA for evaluating and selecting weight vector $\mathbf{w}_{t+1}$ and threshold $\varepsilon$. This framework adaptively selects $\varepsilon$ by balancing real-time tracking accuracy and side performance, achieving performance comparable to the optimal parameter setting, supported by theoretical regret guarantees. Additionally, we propose an efficient method based on the successive convex approximation technique, which significantly reduces time complexity and accelerates computation.

### 3.1 PAS Framework

In this section, we present a novel Passive-Aggressive with Side information (PAS) framework that considers the trade-off between real-time tracking accuracy and side performance. PAS builds upon two variations of PA, each providing closed-form solutions for a given value of $\varepsilon$, without imposing any constraints as follows:

$$\widehat{\mathbf{w}}_{t+1}(\varepsilon) = \begin{cases} \mathbf{w}_t & |\mathbf{w}_t^{\mathsf{T}}\mathbf{x}_t - y_t| \le \varepsilon, \\ \mathbf{w}_t + \text{sign}\left[y_t - \mathbf{w}_t^{\mathsf{T}}\mathbf{x}_t\right]\tau_t\mathbf{x}_t & \text{otherwise,} \end{cases} \tag{2}$$

where

$$\tau_t = \begin{cases} \left(\left|\mathbf{w}_t^{\mathsf{T}}\mathbf{x}_t - y_t\right| - \varepsilon\right)/\|\mathbf{x}_t\|_2^2 & \text{(PA)} \\ \left(\left|\mathbf{w}_t^{\mathsf{T}}\mathbf{x}_t - y_t\right| - \varepsilon\right)/\left(\|\mathbf{x}_t\|_2^2 + \frac{1}{2C}\right) & \text{(PA-II),} \end{cases} \tag{3}$$

and PA-II refers to a robust PA method with an aggressiveness constant $C$. For the regression problem with constraints, the final weight is determined by performing a projection onto the feasible set $\mathcal{W}$:

$$\mathbf{w}_{t+1}(\varepsilon) = \underset{\mathbf{w}\in\mathcal{W}}{\arg\min}\|\mathbf{w} - \widehat{\mathbf{w}}_{t+1}(\varepsilon)\|_2^2. \tag{4}$$

Although Crammer et al. (2006) does not include a projection operation, we demonstrate in Appendix D that PA with lazy projection still achieves a comparable bound to Crammer et al. (2006).

Suppose that at each round $t$, we have a lower semi-continuous convex function $h_t(\mathbf{w})$ that quantifies the performance associated with the side information. To leverage this information and enhance the performance of the weight selection, we integrate $h_t(\mathbf{w})$ into the projection step of the PA method and propose the PAS framework for selecting the next weight vector:

$$\mathbf{w}_{t+1}(\varepsilon) = \underset{\mathbf{w}\in\mathcal{W}}{\arg\min}\left(h_t(\mathbf{w}) + \frac{1}{2\lambda}\|\mathbf{w} - \widehat{\mathbf{w}}_{t+1}(\varepsilon)\|_2^2\right) = \text{prox}_{\lambda h_t}\left(\widehat{\mathbf{w}}_{t+1}\left(\varepsilon\right)\right), \tag{5}$$

where $\text{prox}_{\lambda h_t}$ denotes the proximal operator. In PAS, $\lambda$ serves as the trade-off parameter that quantifies the preference between tracking accuracy and side performance. When $h_t(\mathbf{w})$ is set as a constant, the PAS model essentially simplifies to the original PA method with lazy projection, as shown in Equation (4). By leveraging the proximal operator, we can explicitly integrate side performance into the weight selection process by modifying $h_t(\mathbf{w})$.

From another perspective, $\|\mathbf{w} - \widehat{\mathbf{w}}_{t+1}(\varepsilon)\|_2^2$ can be viewed as a regularization term that passively aligns with the trend of the ground truth. In contrast, $h_t(\mathbf{w})$ serves as the primary loss measurement, acting as the main driver for aggressively updating the weight vector. To understand how the weight vector $\mathbf{w}_{t+1}(\varepsilon)$ is selected, we discuss the following two scenarios:

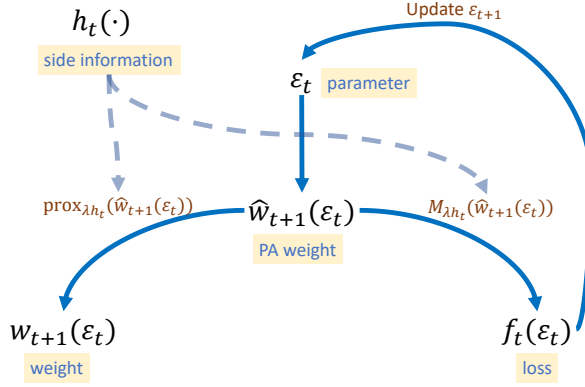

Figure 1: Adaptive learning scheme of APAS.

- If $|\mathbf{w}_t^\mathsf{T}\mathbf{x}_t - y_t| \leq \varepsilon$, we have $\mathbf{w}_{t+1}(\varepsilon) = \arg\min_{\mathbf{w}\in\mathcal{W}} \left(h_t(\mathbf{w}) + \frac{1}{2\lambda}\|\mathbf{w} - \mathbf{w}_t\|_2^2\right)$. This implies that we aim to passively maintain the same weight setting as the previous round while aggressively updating the weight to improve side performance for small real-time tracking errors.

- If $|\mathbf{w}_t^\mathsf{T}\mathbf{x}_t - y_t| > \varepsilon$, we have $\mathbf{w}_{t+1}(\varepsilon) = \arg\min_{\mathbf{w}\in\mathcal{W}} \left(h_t(\mathbf{w}) + \frac{1}{2\lambda}\|\mathbf{w} - \mathbf{w}_t\|_2^2 - \frac{1}{\lambda}\mathbf{w}^\mathsf{T}\mathrm{sign}\left[y_t - \mathbf{w}_t^\mathsf{T}\mathbf{x}_t\right]\tau_t\mathbf{x}_t + \mathrm{const}\right)$. This implies that we aim to passively maintain the same weight setting as the previous round while aggressively updating the weight to improve real-time tracking accuracy and side performance for large real-time tracking errors.

This framework ensures that while the selected weight passively follows the general trend of the data through the $\ell_2$-norm, it actively seeks to improve performance based on the side information, thus achieving a balance between stability and adaptability. Consequently, $\mathbf{w}_{t+1}(\varepsilon)$ corresponds to the point that defines the infimum of the trade-off between side performance $h_t$ and real-time tracking accuracy. The infimum is essentially the Moreau Envelope (Parikh et al., 2014), which we define as the loss function with respect to $\varepsilon$:

$$f_t(\varepsilon) = \inf_{\mathbf{w}\in\mathcal{W}} \left[h_t(\mathbf{w}) + \frac{1}{2\lambda}\|\mathbf{w} - \widehat{\mathbf{w}}_{t+1}(\varepsilon)\|_2^2\right] = M_{\lambda h_t}\left(\widehat{\mathbf{w}}_{t+1}(\varepsilon)\right). \tag{6}$$

Here, $M_{\lambda h_t}$ represents the Moreau Envelope of $\lambda h_t$ with respect to $\lambda\widehat{\mathbf{w}}_{t+1}(\varepsilon)$. In this way, we establish a connection between the determined weight vector $\mathbf{w}_{t+1}(\varepsilon)$ and loss function $f_t(\varepsilon)$ with $\varepsilon$.

### 3.2 Adaptive PAS

The parameter $\varepsilon$ is a crucial component in PAS, as it determines the weight selection $\mathbf{w}_{t+1}(\varepsilon)$ and the performance evaluation $f_t(\varepsilon)$. While the trade-off parameter $\lambda$ has an intuitive interpretation, the process of setting $\varepsilon$ is less straightforward. A smaller threshold $\varepsilon$ may prioritize real-time tracking accuracy but could lead to overfitting, affecting long-term accuracy and compromising side performance. Conversely, a larger $\varepsilon$ might stabilize performance but result in underfitting. Hence, our objective is to develop an adaptive algorithm that can dynamically choose the value of $\varepsilon$ based on the designed loss function $f_t(\varepsilon)$. To facilitate the dynamic selection of $\varepsilon$, we introduce the following assumptions:

**Assumption 1.** *The feasible domain $\mathcal{D}$ of the parameter $\varepsilon$ is bounded with $\mathcal{D} = [\nu, D]$ and $\nu > 0$.*

**Assumption 2.** *The subderivatives of $f_t(\varepsilon)$ is bounded, such that $\sup_{\varepsilon\in\mathcal{D}, t\in[T]} |\partial f_t(\varepsilon)| \leq G$.*

Our proposed adaptive parameter updating scheme for $\mathbf{w}_{t+1}$ and $\varepsilon_{t+1}$ is as follows: At each round $t$, we receive information up to time $t$ and use it to select the next weight vector $\mathbf{w}_{t+1}(\varepsilon_t)$ with $\varepsilon_t$. Subsequently, we update $\varepsilon_{t+1}$ based on the loss $f_t(\varepsilon_t)$. The overall procedure is illustrated in Figure 1. Under Assumptions 1 and 2, the updating rule for $\varepsilon_{t+1}$ is formulated as follows:

$$\varepsilon_{t+1} = \Pi_{\mathcal{D}}\left[\varepsilon_t - \eta_t\tilde{g}_t(\varepsilon_t)\right], \tag{7}$$

where $\Pi_{\mathcal{D}}[\varepsilon] = \min\{\max\{\varepsilon, \nu\}, D\}$, $\eta_t = \frac{\zeta_t \sqrt{D}}{G\sqrt{\nu t}}$, and $\zeta_t = \Pi_{\mathcal{D}}\left[|\mathbf{w}_t^{\mathsf{T}} \mathbf{x}_t - y_t|\right]$. Here, $\tilde{g}_t(\varepsilon)$ is a modified derivative of $f_t(\varepsilon)$, which is defined as follows:

$$\tilde{g}_t(\varepsilon) := \begin{cases} f'_t(\varepsilon) & \text{if } \varepsilon < \zeta_t, \\ \max\{0, \partial_- f_t(\zeta_t)\} & \text{otherwise.} \end{cases} \tag{8}$$

Since $\widehat{\mathbf{w}}_{t+1}(\varepsilon)$ is a continuous piecewise function with respect to $\varepsilon$, being affine on $\varepsilon < \zeta_t$ and constant otherwise, and $M_{\lambda h_t}(\widehat{\mathbf{w}}_{t+1}(\varepsilon))$ is a strongly convex function, their composition $f_t(\varepsilon)$ becomes a piecewise-convex function. Thus, $f_t(\varepsilon)$ is differentiable and strongly convex for $\varepsilon < \zeta_t$, and constant otherwise. The derivative of $f_t(\varepsilon)$ for $\varepsilon < \zeta_t$ can be calculated using the chain rule:

$$f'_t(\varepsilon) = \frac{\partial M_{\lambda h_t}(\widehat{\mathbf{w}}_{t+1}(\varepsilon))}{\partial \varepsilon} = \frac{\partial M_{\lambda h_t}(\widehat{\mathbf{w}}_{t+1}(\varepsilon))}{\partial \widehat{\mathbf{w}}_{t+1}(\varepsilon)} \frac{\partial \widehat{\mathbf{w}}_{t+1}(\varepsilon)}{\partial \varepsilon}. \tag{9}$$

The derivative of the Moreau Envelope $M_{\lambda h_t}$ with respect to $\widehat{\mathbf{w}}_{t+1}(\varepsilon)$ can be calculated as follows:

$$\frac{\partial M_{\lambda h_t}(\widehat{\mathbf{w}}_{t+1}(\varepsilon))}{\partial \widehat{\mathbf{w}}_{t+1}(\varepsilon)} = \frac{1}{\lambda}\left(\widehat{\mathbf{w}}_{t+1}(\varepsilon) - \text{prox}_{\lambda h_t}(\widehat{\mathbf{w}}_{t+1}(\varepsilon))\right). \tag{10}$$

To summarize, the overall updating scheme for $\varepsilon_{t+1}$ and weight vector $\mathbf{w}_{t+1}$ is outlined in Algorithm 1. This adaptive mechanism enables the framework to adjust dynamically to changing environments, eliminating the need for a manually set static threshold. Intuitively, the update process for $\varepsilon$ works as follows: If the real-time tracking error is lower than $\varepsilon_t$ (i.e., $\zeta_t \leq \varepsilon_t$), then according to Equation (8), the derivative $\tilde{g}_t(\varepsilon_t) \geq 0$. Based on the update rule in Equation (7), this suggests that $\varepsilon_t$ is reduced to avoid underestimating the tracking accuracy. Conversely, when the real-time tracking error exceeds $\varepsilon_t$, the update mechanism adjusts $\varepsilon_t$ to strike a balance between minimizing side loss and maintaining tracking accuracy.

---

**Algorithm 1** Adaptive Passive-Aggressive Framework with Side Information (APAS)

---

1: **Input**: trade-off parameter $\lambda$.
2: **Initialize** $\varepsilon_1 \in \mathcal{D}$ and $\mathbf{w}_1 \in \mathcal{W}$.
3: **for** $t = 1, 2, \ldots, T$ **do**
4:     Calculate $\widehat{\mathbf{w}}_{t+1}(\varepsilon_t)$ according to Equation (2);
5:     Update $\mathbf{w}_{t+1}(\varepsilon_t) = \text{prox}_{\lambda h_t}(\widehat{\mathbf{w}}_{t+1}(\varepsilon_t))$;
6:     Update $\varepsilon_{t+1}$ according to Equation (7);
7: **end for**
8: **Output:** $\mathbf{w}_{T+1}(\varepsilon_T)$.

---

### 3.3 Efficient Algorithm

Algorithm 1 requires the calculation of the proximal operator at each iteration (see Line 5), which can be computationally expensive. While this problem can be addressed directly using the Interior Point Method (IPM) with an off-the-shelf solver (Nemirovski, 2004), it generally incurs a high-order time complexity of $O(N^{3.5})$, making it inefficient for large-scale problems.

To improve efficiency, we propose an algorithm that utilizes the Successive Convex Approximation (SCA) framework to accelerate computation (Scutari et al., 2013). SCA reduces time complexity by iteratively optimizing a more manageable surrogate function in place of the original objective function until convergence is reached (Sun et al., 2016; Scutari and Sun, 2018). We denote the objective function of Problem (5) as $u_t(\mathbf{w}) = h_t(\mathbf{w}) + \frac{1}{2\lambda}\|\mathbf{w} - \widehat{\mathbf{w}}_{t+1}(\varepsilon)\|_2^2$. To apply SCA, the surrogate function, denoted as $\tilde{u}_t(\mathbf{w} \mid \mathbf{w}^k)$, should be strongly convex and satisfy the condition that $\nabla \tilde{u}_t(\mathbf{w}^k \mid \mathbf{w}^k) = \nabla u_t(\mathbf{w}^k)$, ensuring the gradients match at $\mathbf{w}^k$.

We employ the first-order Taylor expansion to approximate $h_t(\mathbf{w})$, defining the surrogate function $\tilde{u}_t(\mathbf{w} \mid \mathbf{w}^k)$ as follows:

$$\tilde{u}_t(\mathbf{w} \mid \mathbf{w}^k) = \frac{1}{2\lambda}\mathbf{w}^{\mathsf{T}}\mathbf{w} - \left(\frac{1}{\lambda}\widehat{\mathbf{w}}_{t+1}(\varepsilon) - \nabla h_t(\mathbf{w}^k)\right)^{\mathsf{T}} \mathbf{w} + \text{const.}$$

By simplifying the formulation, we iteratively optimize the following surrogate problem instead:

$$\underset{\mathbf{w}\in\mathbb{R}^N}{\text{minimize}} \quad \|\mathbf{w} - \mathbf{q}^k\|_2^2 \qquad \text{subject to} \quad \mathbf{w} \in \mathcal{W}, \tag{11}$$

where $\mathbf{q}^k = \widehat{\mathbf{w}}_{t+1}(\varepsilon) - \lambda \nabla h_t(\mathbf{w}^k)$. When the feasible set $\mathcal{W}$ exhibits special geometric properties, such as being a probability simplex or a hyperplane, the optimization problem in Equation (11) admits a closed-form solution, as provided in Appendix C (Palomar and Fonollosa, 2005; Duchi et al., 2008).

---

**Algorithm 2** Efficient Algorithm for (5)

---

1: **Input:** $\widehat{\mathbf{w}}_{t+1}(\varepsilon)$, $\lambda$, and $\nabla h_t$.
2: **Initialize** $k = 1$, $\mathbf{w}^1 \in \mathcal{W}$ and $\{\gamma^k\}$.
3: **repeat:**
4:     Solve (11) with $\mathbf{q}^k = \widehat{\mathbf{w}}_{t+1}(\varepsilon) - \lambda \nabla h_t(\mathbf{w}^k)$ and set the optimal point as $\tilde{\mathbf{w}}^{k+1}$;
5:     Compute $\mathbf{w}^{k+1} = \mathbf{w}^k + \gamma^k \left( \tilde{\mathbf{w}}^{k+1} - \mathbf{w}^k \right)$;
6:     $k \leftarrow k + 1$;
7: **until** convergence
8: **Output:** $\mathbf{w}_{t+1}(\varepsilon) = \mathbf{w}^{k+1}$.

---

The overall procedure for efficiently calculating (5) is encapsulated in Algorithm 2. Empirically, this method converges very quickly, reaching the optimal point within only a few iterations. Additionally, it does not require calculating the objective value of $h_t(\mathbf{w})$, making it more flexible for incorporating side information. By setting $\gamma^{k+1} = \gamma^k(1 - \rho\gamma^k)$ with $\rho \in (0, 1)$ and $\gamma^0 < 1/\rho$, Algorithm 2 guarantees convergence to the optimal point of Problem (5). This convergence behavior is analyzed in Proposition 1, with the proof provided in Appendix B.

**Proposition 1.** *With $\gamma^k \in (0, 1]$, $\gamma^k \to 0$ and $\sum_k \gamma^k = +\infty$, Algorithm 2 converges in a finite number of iterations to an optimal solution of (5) or every limit point of the sequence $\{\mathbf{w}^k\}_{k=1}^\infty$ (at least one such point exists) is an optimal solution of (5).*

## 3.4 Regret Analysis

The loss function $f_t(\varepsilon)$ in the APAS framework is piecewise convex, leading to a scenario where it is generally non-convex and non-smooth. In general convex online learning settings, optimal regret bounds are well-established, typically $O(\sqrt{T})$ for $T$ iterations. However, achieving these optimal regret bounds in non-convex online learning scenarios poses significant challenges due to the inherent difficulties in optimizing non-convex functions. Strategies to address these challenges often involve either working with a restricted class of loss functions or focusing on a computationally feasible notion of regret (Hazan et al., 2017; Gao et al., 2018). Additionally, some approaches dealing with general non-convex losses rely on access to sampling oracles, which are impractical in many real-world applications (Maillard and Munos, 2010; Krichene et al., 2015; Agarwal et al., 2019; Suggala and Netrapalli, 2020). Despite recent advances, obtaining optimal regret bounds in non-convex settings remains an open and active area of research.

In our work, we demonstrate that Algorithm 1 can achieve the optimal $O(\sqrt{T})$ regret bound. Our approach is novel in that it does not rely on restrictive assumptions or oracles. Instead, it leverages the properties of the function curve and quasi-convexity, as detailed in Proposition 3 and Proposition 4 in Appendix A. Although $f_t(\varepsilon)$ is non-convex and non-smooth, its behavior along the function curve enables us to derive favorable regret bounds, achieved by carefully designing the learning rate $\eta_t$ and the updating rule for $\varepsilon_{t+1}$. The following theorem formalizes the regret bound of our approach:

**Theorem 2.** *Under Assumptions 1 and 2, Algorithm 1 achieves the following regret bound for $T \geq 1$:*

$$R_T = \sum_{t=1}^{T} f_t(\varepsilon_t) - \min_{\varepsilon \in \mathcal{D}} \sum_{t=1}^{T} f_t(\varepsilon) \leq 2\sqrt{\frac{D^3 G^2}{\nu}}\sqrt{T} = O(\sqrt{T}), \tag{12}$$

*where $D$, $\nu$, and $G$ are constants defined in Assumptions 1 and 2.*

The proof of Theorem 2 is provided in Appendix A. Theorem 2 ensures that Algorithm 1 achieves performance that is comparable to the optimal parameter setting over the long term. Crucially, our approach does not depend on restrictive assumptions or external oracles. Instead, it dynamically adjusts the parameter $\varepsilon$ by responding to real-time changes, allowing for optimal performance in both stable and volatile environments.

# 4 Experiments

To demonstrate the performance of our proposed methods, we conduct experiments using stock lists from S&P 500 and NASDAQ 100 from Yahoo! Finance$^{\text{TM}}$ for an enhanced index-tracking task. Enhanced index tracking is a passive portfolio selection strategy that aims to enhance returns by incorporating tactical tilts towards specific styles, while still maintaining a portfolio that closely mirrors an index (Dose and Cincotti, 2005; Benidis et al., 2017, 2018; Xu et al., 2022).

In our experiments, the instance $\mathbf{x}_t$ represents the stock return at time $t$, where $x_{t,i} = (p_{t,i} - p_{t-1,i})/p_{t-1,i}$ with $p_{t,i}$ denoting the price of asset $i$ at time $t$. The target value $y_t$ is the index return at time $t$. In the enhanced index tracking task, we sequentially select the portfolio weight $\mathbf{w}_t$ at each iteration to mimic the trend of the index $y_t$, where the feasible set is the probability simplex $\mathcal{W} = \{\mathbf{w} \in \mathbb{R}^N \mid \mathbf{1}^\mathsf{T}\mathbf{w} = 1, \mathbf{w} \succeq 0\}$. To achieve a higher return, rather than merely tracking the index, we define the side information as the negative log return, i.e., $h_t(\mathbf{w}) = -\log(1 + \mathbf{x}_t^\mathsf{T}\mathbf{w})$.

We measure the performance of different methods using tracking error and excess cumulative return. The tracking error is quantified by the magnitude of the daily tracking error (MDTE), computed by:

$$\text{Tracking Error} = \frac{1}{T}\sqrt{\sum_{t=1}^{T} \left(\mathbf{w}_t^\mathsf{T}\mathbf{x}_t - y_t\right)^2}. \tag{13}$$

The excess cumulative return is used to assess the performance relative to the tracking index, which represents the discrepancy between the logarithmic cumulative return of the strategy and the index:

$$\text{Excess Cumulative Return} = \sum_{t=1}^{T}\log\left(1 + \mathbf{w}_t^\mathsf{T}\mathbf{x}_t\right) - \sum_{t=1}^{T}\log\left(1 + y_t\right). \tag{14}$$

*Benchmark:* In addition to the base model PA, we compare the performance with two versions of SLAIT: SLAIT-ETE and SLAIT-DR (Benidis et al., 2017). SLAIT-ETE focuses on tracking accuracy, while SLAIT-DR aims to replicate the index while avoiding excessively large drawdowns.

## 4.1 Synthetic Data Experiments

We generate synthetic data by sampling $\mathbf{x}_t \sim \mathcal{N}(\boldsymbol{\mu}, \boldsymbol{\Sigma})$, where $\boldsymbol{\mu} \in \mathbb{R}^N$ and $\boldsymbol{\Sigma} \in \mathbb{R}^{N \times N}$ are the sample mean and sample covariance matrix calculated from the real market data from the S&P 500. The corresponding index value is generated by:

$$y_t = \mathbf{x}_t^\mathsf{T}\mathbf{w}^\star + \omega,$$

where $\omega \sim \mathcal{N}(0, \delta^2)$ represents Gaussian noise, and $\mathbf{w}^\star$ is the true weight of the index components. We generate 50 datasets to test the average performance of different methods, with each dataset containing $T = 200$ observations and $N = 100$ dimensions. The training set consists of 50% of the data, while the test set contains the remaining 50%. Both SLAIT-ETE and SLAIT-DR use a rolling training window of 100-day observations, rebalanced every 3 days.

Figure 2 presents the performance comparison and ablation experiments of the proposed APAS framework against benchmarks on the synthetic dataset. Specifically, Figure 2a illustrates the comparison of excess return and tracking error for APAS and the benchmarks, where the curve for APAS is generated by varying the trade-off parameter $\lambda$. For small $\lambda$, APAS exhibits relatively low tracking error, while for large $\lambda$, APAS achieves higher returns with a slight sacrifice in accuracy. Compared to the benchmarks, APAS demonstrates higher excess cumulative return for the same level of tracking error and lower tracking error for the same level of excess cumulative return.

Figure 2a also shows how varying the trade-off parameter $\lambda$ affects the balance between side performance (measured as excess cumulative return) and tracking error. Generally, $\lambda$ can be selected based on the specific problem's considerations, such as the magnitude of side information and the desired balance between minimizing tracking error and maximizing side performance. In practice, $\lambda$ can be determined using domain knowledge and cross-validation. For example, if a specific range of tracking error is desired, the bisection method can be employed during cross-validation to identify the value of $\lambda$ that maximizes side performance while meeting the tracking error requirement.

Figure 2b compares the performance of the fixed parameter setting with the adaptive one, where PAS refers to the non-adaptive version of APAS with fixed $\varepsilon$. The closer the curve is to the top left, the

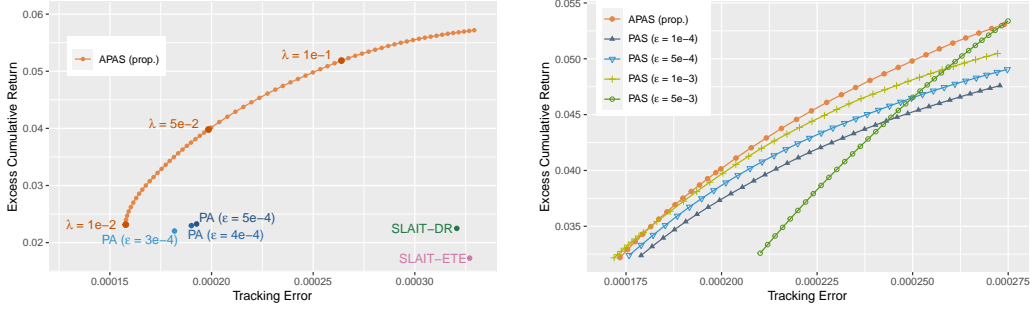

(a) Trade-off between tracking error and excess cumulative return of different methods.

(b) Ablation study: Comparison of PAS with fixed parameters and APAS.

Figure 2: Comparison of tracking error and excess cumulative return on the synthetic dataset.

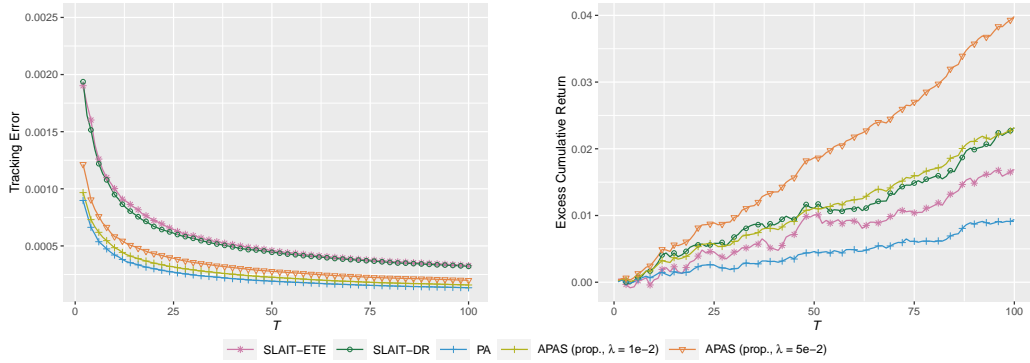

Figure 3: Tracking error and excess cumulative return over time $T$ for different methods on the synthetic dataset.

better the performance. Even without knowing the optimal parameter setting for $\varepsilon$, the adaptive $\varepsilon$ updating scheme in APAS ensures relatively good performance.

We also compare the trends of tracking error and excess cumulative return over time $T$ in Figure 3. This figure shows that both the PA method and the proposed APAS method exhibit relatively low tracking error. Although the PA method has the minimum tracking error, it achieves the lowest excess cumulative return among all methods. In comparison, the APAS method maintains a comparably low tracking error but with a significantly higher excess cumulative return.

It is widely acknowledged that heavy-tailed distributions offer a more realistic model for data-generating processes in financial markets compared to Gaussian distributions (Cardoso et al., 2021, 2022). To further evaluate the performance of APAS in highly volatile and noisy environments, we include a detailed comparison of our proposed methods under various data and noise distributions, available in Appendix E.

## 4.2 Real Market Data Experiments

We conduct simulations on two well-known indices using real market data from Yahoo! Finance™: the S&P 500 Index and the NASDAQ 100 Index. For the S&P 500 Index, we collect data from 2021-01-01 to 2023-01-01, totaling $T = 503$ daily observations with $N = 453$ stocks. For the NASDAQ 100 Index, we collect data from 2019-01-01 to 2021-01-01, also totaling $T = 503$ daily observations with $N = 101$ stocks. For the PA and APAS methods, 50% of the data is used for training, with weights updated adaptively each day based on the latest data. For the SLAIT-ETE and SLAIT-DR methods, the training lookback period is 50% of the data, with rebalancing occurring every 10 days.

Figures 4 and 5 show the performance comparison on the S&P 500 and NASDAQ 100 datasets, respectively. As observed, with a small $\lambda$ setting, APAS has a comparable tracking error to PA while yielding a better excess cumulative return. With a large $\lambda$ setting, APAS exhibits a higher tracking error but achieves the best excess cumulative return among all methods. The real market comparisons across different datasets demonstrate that the proposed APAS model provides a superior trade-off between tracking error and excess cumulative return compared to the benchmarks.

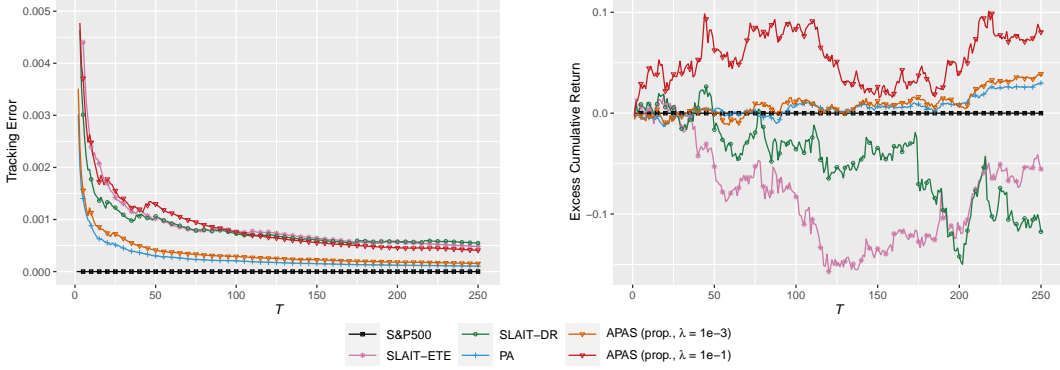

Figure 4: Tracking error and excess cumulative return over time $T$ for different methods on S&P 500 dataset.

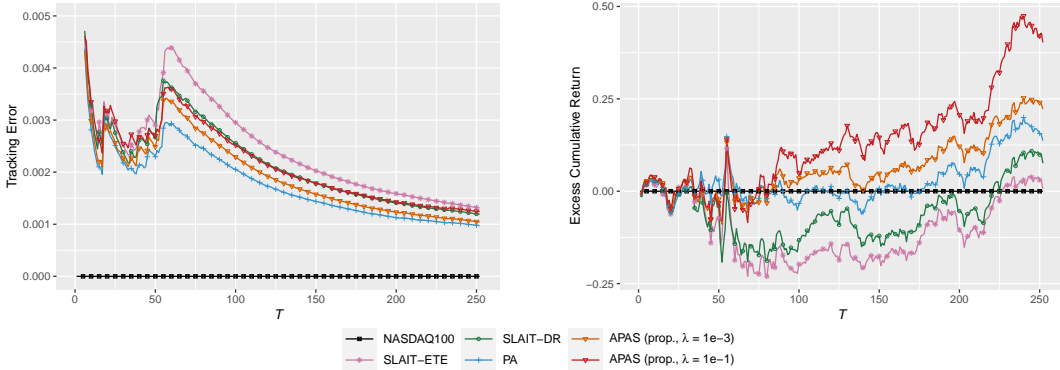

Figure 5: Tracking error and excess cumulative return over time $T$ for different methods on NASDAQ 100 dataset.

## 4.3 Speed Comparison of Acceleration Schemes

This section evaluates the computational efficiency of our proposed method (Algorithm 2) in Section 3.3 across different problem dimensions $N$. The benchmarks include the widely-used convex problem solver CVXR Fu et al. (2020), Projected Gradient Descent (PGD), and Alternating Direction Method of Multipliers (ADMM, Boyd et al., 2011).

We assess the performance of the proposed method over 100 randomized trials, comparing the convergence speed and CPU time (in seconds), as shown in Figure 6. The left panel of Figure 6 illustrates the average convergence gap versus the number of iterations on a dataset with $N = 1000$ dimensions, comparing the proposed method with PGD and ADMM. The right panel displays the average CPU time for each method across different problem dimensions $N$. The results demonstrate that our method converges rapidly to the optimal point, being nearly 100 times faster than CVXR and ADMM and 10 times faster than PGD for high-dimensional data.

To further assess whether time complexity is affected by including different types of side information, we conduct additional experiments using various forms of side information beyond the log return $h_t(\mathbf{w}) = -\log(1 + \mathbf{r}_t^\mathsf{T} \mathbf{w})$, such as:

- Switching cost: $h_t(\mathbf{w}) = ||\mathbf{w} - \mathbf{w}_t||_1$;

- Weighted $\ell_1$ norm: $h_t(\mathbf{w}) = \sum_{i=1}^{N} \rho_i |w_i|$;

- Group Lasso: $h_t(\mathbf{w}) = \sum_{i=1}^{m} \rho_i ||w_{|\mathcal{G}_i}||_2$, where $\mathcal{G}_i, \ldots, \mathcal{G}_m$ are $m$ disjoint groups.

We evaluate the performance of the proposed efficient method with different types of side information functions over 100 randomized trials, comparing the average CPU time (in seconds) in Table 1. From Table 1, it appears that group Lasso incurs higher CPU times, especially for larger dimensions $N$, due to the added complexity of calculating norms for disjoint groups. In general, while the type of side information can impact the computational time, the APAS framework maintains efficiency across different scenarios.

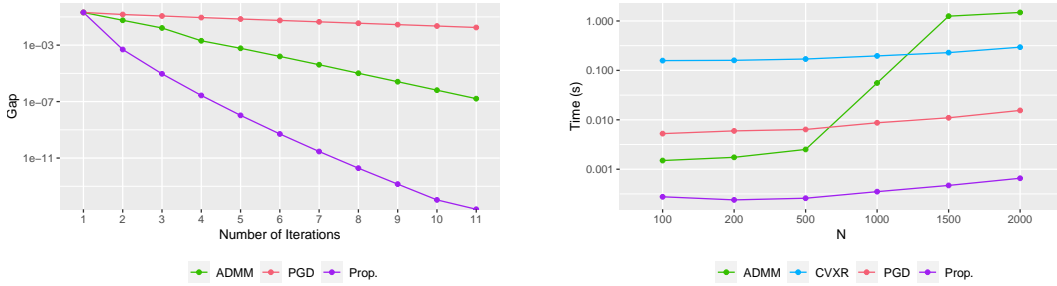

Figure 6: Average convergence speed and CPU time comparison of 100 randomized trials on $N$-dimensional datasets of Algorithm 2.

|  | log return | switching cost | weighted $\ell_1$ norm | group Lasso |
|---|---|---|---|---|
| $N = 500$ | $0.00084 \pm 0.00051$ | $0.00084 \pm 0.00163$ | $0.00045 \pm 0.00051$ | $0.00162 \pm 0.00054$ |
| $N = 1000$ | $0.00119 \pm 0.00050$ | $0.00084 \pm 0.00048$ | $0.00084 \pm 0.00052$ | $0.00252 \pm 0.00154$ |
| $N = 2000$ | $0.00181 \pm 0.00103$ | $0.00156 \pm 0.00129$ | $0.00113 \pm 0.00043$ | $0.00344 \pm 0.00138$ |
| $N = 5000$ | $0.00335 \pm 0.00080$ | $0.00356 \pm 0.00125$ | $0.00282 \pm 0.00122$ | $0.00702 \pm 0.00205$ |

Table 1: Average CPU time (in seconds) for different side information functions over 100 randomized trials of Algorithm 2.

## 5    Conclusions

In this paper, we addressed the limitations of the Passive-Aggressive (PA) algorithm in online regression, particularly in determining the appropriate threshold and integrating side information for weight selection. To tackle these issues, we proposed the APAS framework, which incorporates side information into PA. Our APAS framework adaptively selects the threshold parameter, enabling it to leverage side information for improved performance while maintaining a low tracking error. We demonstrated the robustness and effectiveness of APAS through an $O(\sqrt{T})$ regret bound, even with non-convex loss functions. Additionally, we developed an efficient algorithm that significantly reduced computational complexity without compromising theoretical performance guarantees. Comprehensive experiments on synthetic and real market datasets validated the effectiveness and efficiency of APAS, highlighting its practical applicability across various scenarios.

## Acknowledgments and Disclosure of Funding

We would like to thank the anonymous reviewers for their helpful comments. This work was supported by the Hong Kong GRF 16206123 research grant and the Hong Kong RGC Postdoctoral Fellowship Scheme of Project No. PDFS2425-6S05.

# References

Agarwal, N., Gonen, A., and Hazan, E. (2019). Learning in non-convex games with an optimization oracle. In *Conference on Learning Theory*, pages 18–29. PMLR.

Anava, O., Hazan, E., Mannor, S., and Shamir, O. (2013). Online learning for time series prediction. In *Conference on Learning Theory*, pages 172–184. PMLR.

Anava, O., Hazan, E., and Zeevi, A. (2015). Online time series prediction with missing data. In *International Conference on Machine Learning*, pages 2191–2199. PMLR.

Benidis, K., Feng, Y., and Palomar, D. P. (2017). Sparse portfolios for high-dimensional financial index tracking. *IEEE Transactions on Signal Processing*, 66(1):155–170.

Benidis, K., Feng, Y., Palomar, D. P., et al. (2018). Optimization methods for financial index tracking: From theory to practice. *Foundations and Trends® in Optimization*, 3(3):171–279.

Boyd, S., Parikh, N., Chu, E., Peleato, B., Eckstein, J., et al. (2011). Distributed optimization and statistical learning via the alternating direction method of multipliers. *Foundations and Trends® in Machine learning*, 3(1):1–122.

Cardoso, J. V. d. M., Ying, J., and Palomar, D. P. (2021). Graphical models in heavy-tailed markets. In *Advances in Neural Information Processing Systems*, volume 34, pages 19989–20001.

Cardoso, J. V. d. M., Ying, J., and Palomar, D. P. (2022). Learning bipartite graphs: Heavy tails and multiple components. In *Advances in Neural Information Processing Systems*, volume 35, pages 14044–14057.

Crammer, K., Dekel, O., Keshet, J., Shalev-Shwartz, S., and Singer, Y. (2006). Online passive-aggressive algorithms. *Journal of Machine Learning Research*, 7(19):551–585.

Dose, C. and Cincotti, S. (2005). Clustering of financial time series with application to index and enhanced index tracking portfolio. *Physica A: Statistical Mechanics and its Applications*, 355(1):145–151.

Duchi, J., Hazan, E., and Singer, Y. (2011). Adaptive subgradient methods for online learning and stochastic optimization. *Journal of Machine Learning Research*, 12(7).

Duchi, J., Shalev-Shwartz, S., Singer, Y., and Chandra, T. (2008). Efficient projections onto the $\ell_1$-ball for learning in high dimensions. In *International Conference on Machine Learning*, pages 272–279.

Duchi, J. C., Shalev-Shwartz, S., Singer, Y., and Tewari, A. (2010). Composite objective mirror descent. In *Conference on Learning Theory*, volume 10, pages 14–26.

Fu, A., Narasimhan, B., and Boyd, S. (2020). CVXR: An R package for disciplined convex optimization. *Journal of Statistical Software*, 94(14):1–34.

Gao, X., Li, X., and Zhang, S. (2018). Online learning with non-convex losses and non-stationary regret. In *International Conference on Artificial Intelligence and Statistics*, pages 235–243. PMLR.

Hazan, E. (2022). *Introduction to online convex optimization*. MIT Press.

Hazan, E., Agarwal, A., and Kale, S. (2007). Logarithmic regret algorithms for online convex optimization. *Machine Learning*, 69(2):169–192.

Hazan, E., Lee, H., Singh, K., Zhang, C., and Zhang, Y. (2018). Spectral filtering for general linear dynamical systems. *Advances in Neural Information Processing Systems*, 31.

Hazan, E. and Seshadhri, C. (2007). Adaptive algorithms for online decision problems. In *Electronic Colloquium on Computational Complexity*, volume 14.

Hazan, E. and Seshadhri, C. (2009). Efficient learning algorithms for changing environments. In *International Conference on Machine Learning*, pages 393–400.

Hazan, E., Singh, K., and Zhang, C. (2017). Efficient regret minimization in non-convex games. In *International Conference on Machine Learning*, pages 1433–1441. PMLR.

Herbster, M. (2001). Learning additive models online with fast evaluating kernels. In *Computational Learning Theory: 14th Annual Conference on Computational Learning Theory, COLT 2001 and 5th European Conference on Computational Learning Theory, EuroCOLT 2001 Amsterdam, The Netherlands, July 16–19, 2001 Proceedings 14*, pages 444–460. Springer.

Krichene, W., Balandat, M., Tomlin, C., and Bayen, A. (2015). The hedge algorithm on a continuum. In *International Conference on Machine Learning*, pages 824–832. PMLR.

Lale, S., Azizzadenesheli, K., Hassibi, B., and Anandkumar, A. (2020). Logarithmic regret bound in partially observable linear dynamical systems. *Advances in Neural Information Processing Systems*, 33:20876–20888.

Li, B. and Hoi, S. C. (2012). On-line portfolio selection with moving average reversion. In *International Conference on Machine Learning*, pages 563–570.

Li, B., Zhao, P., Hoi, S. C., and Gopalkrishnan, V. (2012). PAMR: Passive aggressive mean reversion strategy for portfolio selection. *Machine Learning*, 87:221–258.

Ma, J., Saul, L. K., Savage, S., and Voelker, G. M. (2009). Identifying suspicious urls: an application of large-scale online learning. In *International Conference on Machine Learning*, pages 681–688.

Maillard, O.-A. and Munos, R. (2010). Online learning in adversarial lipschitz environments. In *Joint European Conference on Machine Learning and Knowledge Discovery in Databases*, pages 305–320. Springer.

Nemirovski, A. (2004). Interior point polynomial time methods in convex programming. *Lecture notes*, 42(16):3215–3224.

Orabona, F. (2019). A modern introduction to online learning. *arXiv preprint arXiv:1912.13213*.

Palomar, D. P. and Fonollosa, J. R. (2005). Practical algorithms for a family of waterfilling solutions. *IEEE Transactions on Signal Processing*, 53(2):686–695.

Parikh, N., Boyd, S., et al. (2014). Proximal algorithms. *Foundations and trends® in Optimization*, 1(3):127–239.

Scutari, G., Facchinei, F., Song, P., Palomar, D. P., and Pang, J.-S. (2013). Decomposition by partial linearization: Parallel optimization of multi-agent systems. *IEEE Transactions on Signal Processing*, 62(3):641–656.

Scutari, G. and Sun, Y. (2018). Parallel and distributed successive convex approximation methods for big-data optimization. *Lecture Notes in Mathematics, C.I.M.E, Springer Verlag series*.

Shalev-Shwartz, S. and Ben-David, S. (2014). *Understanding machine learning: from theory to algorithms*. Cambridge University Press, USA.

Shalev-Shwartz, S. et al. (2012). Online learning and online convex optimization. *Foundations and Trends® in Machine Learning*, 4(2):107–194.

Suggala, A. S. and Netrapalli, P. (2020). Online non-convex learning: Following the perturbed leader is optimal. In *Algorithmic Learning Theory*, pages 845–861. PMLR.

Sun, Y., Babu, P., and Palomar, D. P. (2016). Majorization-minimization algorithms in signal processing, communications, and machine learning. *IEEE Transactions on Signal Processing*, 65(3):794–816.

Tsiamis, A. and Pappas, G. J. (2022). Online learning of the kalman filter with logarithmic regret. *IEEE Transactions on Automatic Control*, 68(5):2774–2789.

Van Erven, T. and Koolen, W. M. (2016). Metagrad: Multiple learning rates in online learning. *Advances in Neural Information Processing Systems*, 29.

Xu, F., Ma, J., and Lu, H. (2022). Group sparse enhanced indexation model with adaptive beta value. *Quantitative Finance*, 22(10):1905–1926.

Zhang, Z., Cutkosky, A., and Paschalidis, Y. (2024). Unconstrained dynamic regret via sparse coding. *Advances in Neural Information Processing Systems*, 36.

Zhao, P. and Hoi, S. C. (2013). Cost-sensitive online active learning with application to malicious url detection. In *Proceedings of the 19th ACM SIGKDD International Conference on Knowledge Discovery and Data Mining*, pages 919–927.

Zinkevich, M. (2003). Online convex programming and generalized infinitesimal gradient ascent. In *International Conference on Machine Learning*, pages 928–936.

# Appendix

In the following sections, we present the theoretical proofs for Theorem 2 and Proposition 1. Additionally, we provide closed-form solutions for Algorithm 2 under special cases not explicitly stated in the main manuscript, along with a detailed specification of the regret bound analysis for the Passive-Aggressive (PA) method with lazy projection. Furthermore, we include additional experiments to assess the robustness of the proposed APAS framework under various conditions.

## A  Proof of Theorem 2

The proof of Theorem 2 relies on the first order bound of $f_t(\varepsilon)$, shown in the following proposition.

**Proposition 3.** *Under Assumption 1 and 2, $f_t(\varepsilon)$ is quasi-convex on $\mathcal{D} = [\nu, D]$. With the definition of $\tilde{g}_t(\varepsilon)$ in Equation (8) and $\zeta_t = \Pi_{\mathcal{D}}\left[|\mathbf{w}_t^{\mathsf{T}}\mathbf{x}_t - y_t|\right]$, for all $t \in [T]$ and all $v, u \in \mathcal{D}$, we have*

$$f_t(v) - f_t(u) \le \tilde{g}_t(v)(v - \tilde{u}), \tag{15}$$

*where $\tilde{u} = \min\{u, \zeta_t\}$.*

The proof of Proposition 3 is detailed in Appendix A.1. Let $\varepsilon^\star \in \arg\min_{\varepsilon \in \mathcal{D}} \sum_{t=1}^{T} f_t(\varepsilon)$. According to Proposition 3, we have:

$$f_t(\varepsilon_t) - f_t(\varepsilon^\star) \le \tilde{g}_t(\varepsilon_t)(\varepsilon_t - z_t),$$

where $z_t = \min\{\zeta_t, \varepsilon^\star\}$. Since $\varepsilon_{t+1} = \Pi_{\mathcal{D}}\left[\varepsilon_t - \eta_t \tilde{g}_t(\varepsilon_t)\right]$ and employing the Pythagorean theorem, we have:

$$(\varepsilon_{t+1} - z_t)^2 = (\Pi_{\mathcal{D}}\left[\varepsilon_t - \eta_t \tilde{g}_t(\varepsilon_t)\right] - z_t)^2 \le (\varepsilon_t - \eta_t \tilde{g}_t(\varepsilon_t) - z_t)^2.$$

By properly reformulating the inequality, we have:

$$\tilde{g}_t(\varepsilon_t)(\varepsilon_t - z_t) \le \phi_t(z_t) - \psi_t(z_t) + \frac{\eta_t G^2}{2},$$

where $\phi_t(z_t) = \frac{\varepsilon_t^2 - 2\varepsilon_t z_t}{2\eta_t}$ and $\psi_t(z_t) = \frac{\varepsilon_{t+1}^2 - 2\varepsilon_{t+1} z_t}{2\eta_t}$. Summing from $t = 1$ to $T$, we have:

$$
\begin{aligned}
R_T &= \sum_{t=1}^{T} \left(f_t\left(\varepsilon_t\right) - f_t\left(\varepsilon^\star\right)\right) \\
&\le \sum_{t=1}^{T} \left(\phi_t(z_t) - \psi_t(z_t) + \frac{\eta_t G^2}{2}\right) \\
&= \phi_1(z_1) - \psi_T(z_T) + \sum_{t=2}^{T} \left(\phi_t(z_t) - \psi_{t-1}(z_{t-1})\right) + \frac{G^2}{2} \sum_{t=1}^{T} \eta_t.
\end{aligned}
$$

Thus, we only need to bound $\phi_t(z_t) - \psi_{t-1}(z_{t-1})$.

**Proposition 4.** *Set $\eta_t = \frac{\zeta_t \sqrt{D}}{G\sqrt{\nu t}}$ with $\zeta_t = \Pi_{\mathcal{D}}\left[|\mathbf{w}_t^{\mathsf{T}}\mathbf{x}_t - y_t|\right]$. Under Assumptions 1 and 2, for $\varepsilon^\star \in \arg\min_{\varepsilon \in \mathcal{D}} \sum_{t=1}^{T} f_t(\varepsilon)$, and $z_t = \min\{\zeta_t, \varepsilon^\star\}$, we have:*

$$\phi_t(z_t) - \psi_{t-1}(z_{t-1}) \le \frac{D^2}{2}\left(\frac{1}{\eta_t} - \frac{1}{\eta_{t-1}}\right), \tag{16}$$

*where $\phi_t(z_t) = \frac{\varepsilon_t^2 - 2\varepsilon_t z_t}{2\eta_t}$ and $\psi_t(z_t) = \frac{\varepsilon_{t+1}^2 - 2\varepsilon_{t+1} z_t}{2\eta_t}$.*

The proof for Proposition 4 is detailed in Appendix A.2. Based on Proposition 4, we have

$$R_T \le \frac{D^2}{\eta_T} + \frac{G^2}{2} \sum_{t=1}^{T} \eta_t \le 2\sqrt{\frac{D^3 G^2}{\nu}} \sqrt{T} = O(\sqrt{T}).$$

### A.1 Proof of Proposition 3

*Proof.* First we show that $f_t(\varepsilon)$ is quasi-convex on $\mathcal{D}$. The loss function $f_t(\varepsilon)$ is the Moreau Envelop of $\widehat{\mathbf{w}}_{t+1}(\varepsilon)$, which is given by:

$$f_t(\varepsilon) = M_{\lambda h_t}\left(\widehat{\mathbf{w}}_{t+1}(\varepsilon)\right) = \inf_{\mathbf{w}\in\mathcal{W}}\left[h_t(\mathbf{w}) + \frac{1}{2\lambda}\|\mathbf{w} - \widehat{\mathbf{w}}_{t+1}(\varepsilon)\|_2^2\right].$$

Here, $M_{\lambda h_t}\left(\widehat{\mathbf{w}}_{t+1}(\varepsilon)\right)$ is strongly convex and smooth with respect to $\widehat{\mathbf{w}}_{t+1}(\varepsilon)$. Furthermore, $\widehat{\mathbf{w}}_{t+1}(\varepsilon)$ is a piecewise continuous affine function of $\varepsilon$, as shown in Equation (2). It is constant if $\varepsilon \geq |\mathbf{w}_t^\mathsf{T}\mathbf{x}_t - y_t|$ and an affine function of $\varepsilon$ otherwise. Since $f_t(\varepsilon)$ is a composite function of the strongly convex function $M_{\lambda h_t}\left(\widehat{\mathbf{w}}_{t+1}(\varepsilon)\right)$ and the piecewise continuous affine function $\widehat{\mathbf{w}}_{t+1}(\varepsilon)$, we have:

$$f_t(\varepsilon) = \begin{cases} \text{strongly convex function} & \varepsilon \in [\nu, \zeta_t) \\ \text{const} & \varepsilon \in [\zeta_t, D], \end{cases}$$

where $\zeta_t = \Pi_{\mathcal{D}}\left[|\mathbf{w}_t^\mathsf{T}\mathbf{x}_t - y_t|\right]$. Thus, it is straightforward to verify that $f_t(\varepsilon)$ is quasi-convex.

To verify the inequality (15), we analyze different cases. First, we consider the simplest case where $\zeta_t = \nu$, which implies that $|\mathbf{w}_t^\mathsf{T}\mathbf{x}_t - y_t| \leq \nu$ and $f_t(\varepsilon)$ is a constant on $\mathcal{D} = [\nu, D]$. Since $\tilde{g}_t(\varepsilon) \geq 0$ according to (8) and $\tilde{u} = \nu$, it is straightforward to verify that:

$$f_t(v) - f_t(u) = 0 \leq \tilde{g}_t(v)(v - \tilde{u}).$$

Then, we consider the case where $\zeta_t = D$, which implies that $|\mathbf{w}_t^\mathsf{T}\mathbf{x}_t - y_t| \geq D$ and $f_t(\varepsilon)$ is strongly convex on $\mathcal{D} = [\nu, D]$. Here, $\tilde{u} = u$, and we consider the following cases:

1. For $v < \zeta_t$, we have $\tilde{g}_t(v) = f_t'(v)$, and thus, by convexity:
$$f_t(v) - f_t(u) \leq f_t'(v)(v - u) = \tilde{g}_t(v)(v - \tilde{u}).$$

2. For $v = \zeta_t$: if $\partial_- f_t(v) \geq 0$, then $\tilde{g}_t(v) = \partial_- f_t(v)$, and by convexity:
$$f_t(v) - f_t(u) \leq \partial_- f_t(v)(v - u) = \tilde{g}_t(v)(v - \tilde{u}).$$

If $\partial_- f_t(v) < 0$, then $\tilde{g}_t(v) = 0$, and we have:
$$f_t(v) - f_t(u) \leq \partial_- f_t(v)(v - u) \leq 0 = \tilde{g}_t(v)(v - \tilde{u}).$$

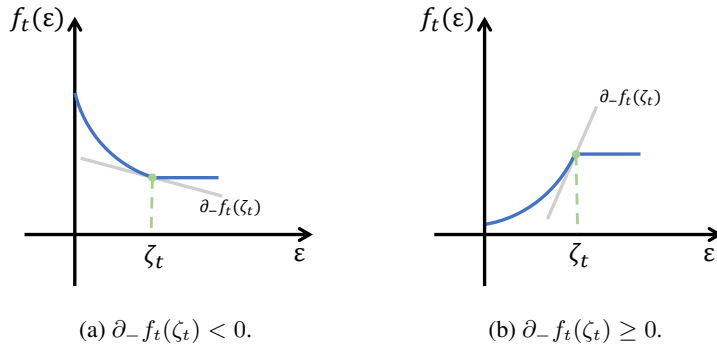

(a) $\partial_- f_t(\zeta_t) < 0$.      (b) $\partial_- f_t(\zeta_t) \geq 0$.

Figure 7: Illustration for curves of $f_t(\varepsilon)$ with $\nu < \zeta_t < D$.

Next, we consider the case where $\nu < \zeta_t < D$, meaning that $\zeta_t = |\mathbf{w}_t^\mathsf{T}\mathbf{x}_t - y_t|$. Figure 7 illustrates the curve of $f_t(\varepsilon)$. The curve of the loss function $\tilde{f}_t(\varepsilon)$ could be divided into two categories: when $\partial_- f_t(\zeta_t) < 0$, we obtain a convex function, as shown in Figure 7a; when $\partial_- f_t(\zeta_t) \geq 0$, we get a quasi-convex function, as shown in Figure 7b. To verify the inequalities (15), we consider the following cases:

1. For $\nu \leq v < \zeta_t$, we have $\tilde{g}_t(v) = f_t'(v)$:

(a) If $\nu \leq u < \zeta_t$, then $\tilde{u} = \min\{u, \zeta_t\} = u$. We can directly verify inequality (15) directly by convexity:

$$f_t(v) - f_t(u) \leq f_t'(v)(v - u) = \tilde{g}_t(v)(v - \tilde{u}).$$

(b) If $\zeta_t \leq u \leq D$, then $\tilde{u} = \min\{u, \zeta_t\} = \zeta_t$. By convexity, we have:

$$f_t(v) - f_t(u) = f_t(v) - f_t(\zeta_t) \leq f_t'(v)(v - \zeta_t) = \tilde{g}_t(v)(v - \tilde{u}).$$

2. For $\zeta_t \leq v \leq D$, we have $\tilde{g}_t(v) = \max\{0, \partial_- f_t(\zeta_t)\}$:

(a) If $\nu \leq u < \zeta_t$ and $\partial_- f_t(\zeta_t) > 0$, then $\tilde{g}_t(v) = \partial_- f_t(\zeta_t)$. Thus, by convexity:

$$f_t(v) - f_t(u) = f_t(\zeta_t) - f_t(u) \leq \partial_- f_t(\zeta_t)(\zeta_t - u) \leq \partial_- f_t(\zeta_t)(v - u) = \tilde{g}_t(v)(v - \tilde{u}).$$

If $\nu \leq u < \zeta_t$ and $\partial_- f_t(\zeta_t) \leq 0$, then $\tilde{g}_t(v) = 0$. Since $f_t(\varepsilon)$ is strongly convex on $[\nu, \zeta_t]$, we have $f_t(u) > f_t(\zeta_t)$. Thus, we have:

$$f_t(v) - f_t(u) = f_t(\zeta_t) - f_t(u) < 0 = \tilde{g}_t(v)(v - u) = \tilde{g}_t(v)(v - \tilde{u}).$$

(b) If $\zeta_t \leq u \leq D$, it is straightforward to verify that $f_t(v) - f_t(u) = 0$ and $\tilde{g}_t(v)(v - \zeta_t) \geq 0$. Then we have:

$$f_t(v) - f_t(u) = 0 \leq \tilde{g}_t(v)(v - \zeta_t) = \tilde{g}_t(v)(v - \tilde{u}).$$

Thus, we prove inequality (15). $\qquad\square$

### A.2 Proof of Proposition 4

*Proof.* Let $\phi_t(z_t) = \frac{\varepsilon_t^2 - 2\varepsilon_t z_t}{2\eta_t}$ and $\psi_t(z_t) = \frac{\varepsilon_{t+1}^2 - 2\varepsilon_{t+1} z_t}{2\eta_t}$, where $\varepsilon^\star \in \arg\min_{\varepsilon \in \mathcal{D}} \sum_{t=1}^{T} f_t(\varepsilon)$ and $z_t = \min\{\zeta_t, \varepsilon^\star\}$. Let $\zeta_t = \Pi_{\mathcal{D}}\left[|\mathbf{w}_t^\mathsf{T}\mathbf{x}_t - y_t|\right]$, and consider the following four situations for $\eta_t = \frac{\zeta_t \sqrt{D}}{G\sqrt{\nu t}}$:

- if $\varepsilon^\star \geq \zeta_{t-1}$ and $\varepsilon^\star \geq \zeta_t$:

$$
\begin{aligned}
\phi_t(z_t) - \psi_{t-1}(z_{t-1}) &= \phi_t(\zeta_t) - \psi_{t-1}(\zeta_{t-1}) \\
&= \frac{\varepsilon_t^2 - 2\varepsilon_t \zeta_t}{2\eta_t} - \frac{\varepsilon_t^2 - 2\varepsilon_t \zeta_{t-1}}{2\eta_{t-1}} \\
&\leq \frac{D^2}{2}\left(\frac{1}{\eta_t} - \frac{1}{\eta_{t-1}}\right) + \varepsilon_t\left(\frac{\zeta_{t-1}}{\eta_{t-1}} - \frac{\zeta_t}{\eta_t}\right) \\
&= \frac{D^2}{2}\left(\frac{1}{\eta_t} - \frac{1}{\eta_{t-1}}\right) + \frac{G\sqrt{\nu}\varepsilon_t}{\sqrt{D}}\left(\sqrt{t-1} - \sqrt{t}\right) \\
&\leq \frac{D^2}{2}\left(\frac{1}{\eta_t} - \frac{1}{\eta_{t-1}}\right).
\end{aligned}
$$

- if $\varepsilon^\star \geq \zeta_{t-1}$ and $\varepsilon^\star < \zeta_t$:

$$
\begin{aligned}
\phi_t(z_t) - \psi_{t-1}(z_{t-1}) &= \phi_t(\varepsilon^\star) - \psi_{t-1}(\zeta_{t-1}) \\
&= \frac{\varepsilon_t^2 - 2\varepsilon_t\varepsilon^\star}{2\eta_t} - \frac{\varepsilon_t^2 - 2\varepsilon_t\zeta_{t-1}}{2\eta_{t-1}} \\
&\leq \frac{\varepsilon_t^2 - 2\varepsilon_t\varepsilon^\star}{2\eta_t} - \frac{\varepsilon_t^2 - 2\varepsilon_t\varepsilon^\star}{2\eta_{t-1}} \qquad \text{[Since } \varepsilon^\star \geq \zeta_{t-1}] \\
&\leq \frac{D^2}{2}\left(\frac{1}{\eta_t} - \frac{1}{\eta_{t-1}}\right).
\end{aligned}
$$

- if $\varepsilon^\star < \zeta_{t-1}$ and $\varepsilon^\star \geq \zeta_t$:

$$\phi_t(z_t) - \psi_{t-1}(z_{t-1}) = \phi_t(\zeta_t) - \psi_{t-1}(\varepsilon^\star)$$

$$= \frac{\varepsilon_t^2 - 2\varepsilon_t\zeta_t}{2\eta_t} - \frac{\varepsilon_t^2 - 2\varepsilon_t\varepsilon^\star}{2\eta_{t-1}}$$

$$\leq \frac{\varepsilon_t^2 - 2\varepsilon_t\zeta_t}{2\eta_t} - \frac{\varepsilon_t^2 - 2\varepsilon_t\zeta_{t-1}}{2\eta_{t-1}} \qquad \text{[Since } \varepsilon^\star < \zeta_{t-1}\text{]}$$

$$\leq \frac{D^2}{2}\left(\frac{1}{\eta_t} - \frac{1}{\eta_{t-1}}\right).$$

- if $\varepsilon^\star < \zeta_{t-1}$ and $\varepsilon^\star < \zeta_t$:

$$\phi_t(z_t) - \psi_{t-1}(z_{t-1}) = \phi_t(\varepsilon^\star) - \psi_{t-1}(\varepsilon^\star)$$

$$= \frac{\varepsilon_t^2 - 2\varepsilon_t\varepsilon^\star}{2\eta_t} - \frac{\varepsilon_t^2 - 2\varepsilon_t\varepsilon^\star}{2\eta_{t-1}}$$

$$\leq \frac{D^2}{2}\left(\frac{1}{\eta_t} - \frac{1}{\eta_{t-1}}\right).$$

To summarize, we have

$$\phi_t(z_t) - \psi_{t-1}(z_{t-1}) \leq \frac{D^2}{2}\left(\frac{1}{\eta_t} - \frac{1}{\eta_{t-1}}\right).$$

$\square$

## B  Proof of Proposition 1

*Proof.* Since (Scutari et al., 2013, Assumptions A1-A4) hold and (5) is a convex problem, the proof for Proposition (1) follows directly from (Scutari et al., 2013, Theorem 3). $\square$

## C  Efficient Euclidean Projection Methods

### C.1  Projection onto the Probability Simplex

**Proposition 5** (Projection onto Simplex (Palomar and Fonollosa, 2005)). *When* $\mathcal{W} = \{\mathbf{w} \in \mathbb{R}^N \mid \mathbf{1}^\mathsf{T}\mathbf{w} = 1, \mathbf{w} \succeq 0\}$, *problem (11) has a closed-form solution given by:*

$$w_i^\star = \left[q_i^k + \kappa\right]_+ \quad i = 1, \ldots, N, \tag{17}$$

*where* $\kappa = \frac{1}{\rho}\left(1 - \sum_{i=1}^{\rho} q_{[i]}^k\right)$ *with* $\rho = \max\left\{1 \leq j \leq N : q_{[j]}^k + \frac{1}{j}\left(1 - \sum_{i=1}^{j} q_{[i]}^k\right) > 0\right\}$, *and* $q_{[i]}^k$ *are the sorted elements of* $\mathbf{q}^k$, *arranged such that* $q_{[1]}^k \geq q_{[2]}^k \geq \cdots \geq q_{[N]}^k$.

### C.2  Projection onto $\ell_1$ Norm Ball

**Proposition 6** (Projection onto $\ell_1$ Norm Ball (Duchi et al., 2008)). *When* $\mathcal{W} = \{\mathbf{w} \in \mathbb{R}^N \mid \|\mathbf{w}\|_1 \leq c\}$ *for some constant* $c > 0$, *problem (11) has a closed-form solution given by:*

$$w_i^\star = sign(q_i^k)\left[\left|q_i^k\right| - \tau\right]_+ \quad i = 1, \ldots, N, \tag{18}$$

*where* $\tau$ *is chosen such that* $\sum_{i=1}^{N}\left[\left|q_i^k\right| - \tau\right]_+ = c$. *The value of* $\tau$ *can be efficiently found by sorting* $|q_i^k|$ *and using a bisection search.*

## D  Regret Analysis of PA with Lazy Projection

**Lemma 7.** *Let* $(\mathbf{x}_1, y_1), \ldots, (\mathbf{x}_T, y_T)$ *be an arbitrary sequence of examples, where* $\mathbf{x}_t \in \mathbb{R}^N$ *and* $y_t \in \mathbb{R}$ *for all t. Let* $\xi_t = 0$ *for* $|\mathbf{w}_t^\mathsf{T}\mathbf{x}_t - y_t| \leq \varepsilon$ *and* $\xi_t = \tau_t$ *in Equation (3) otherwise. Let*

$\mathcal{W} \subseteq \mathbb{R}^N$ *be the feasible set of the weight vector and* $\mathbf{u}$ *be an arbitrary vector in* $\mathcal{W}$. *Define* $\ell_t = \ell_\varepsilon(\mathbf{w}_t; (\mathbf{x}_t, y_t))$ *and* $\ell_t^\star = \ell_\varepsilon(\mathbf{u}; (\mathbf{x}_t, y_t))$. *The following bound holds for any* $\mathbf{u} \in \mathbb{R}^N$:

$$\sum_{t=1}^{T} \xi_t \left( 2\ell_t - \xi_t \|\mathbf{x}_t\|_2^2 - 2\ell_t^\star \right) \leq \|\mathbf{u}\|_2^2. \tag{19}$$

*Proof.* The proof is mainly based on (Crammer et al., 2006, Lemma 6) with minor modification. To facilitate the analysis of the regret bound, we rewrite the recursive updating rule of $\widehat{\mathbf{w}}_{t+1}$ in PA as

$$\widehat{\mathbf{w}}_{t+1} = \mathbf{w}_t + \text{sign}\left[y_t - \mathbf{w}_t^\mathsf{T}\mathbf{x}_t\right]\xi_t \mathbf{x}_t,$$

where $\xi_t = 0$ for $|\mathbf{w}_t^\mathsf{T}\mathbf{x}_t - y_t| \leq \varepsilon$ and $\xi_t = \tau_t$ in Equation (3) otherwise. By projecting on the feasible set $\mathcal{W}$, we have the weight generated by PA with lazy projection as the following:

$$\mathbf{w}_{t+1} = \Pi_{\mathcal{W}}(\widehat{\mathbf{w}}_{t+1}) = \underset{\mathbf{w} \in \mathcal{W}}{\arg\min} \|\mathbf{w} - \widehat{\mathbf{w}}_{t+1}\|_2^2.$$

Without loss of generality, we set $\widehat{\mathbf{w}}_1 = \mathbf{0}$ and define

$$\Delta_t = \|\mathbf{w}_t - \mathbf{u}\|_2^2 - \|\mathbf{w}_{t+1} - \mathbf{u}\|_2^2.$$

Summing from 1 to $T$, we have

$$
\begin{aligned}
\sum_{t=1}^{T} \Delta_t &= \sum_{t=1}^{T} \left( \|\mathbf{w}_t - \mathbf{u}\|_2^2 - \|\mathbf{w}_{t+1} - \mathbf{u}\|_2^2 \right) \\
&= \|\mathbf{w}_1 - \mathbf{u}\|_2^2 - \|\mathbf{w}_{T+1} - \mathbf{u}\|_2^2 \\
&\leq \|\mathbf{w}_1 - \mathbf{u}\|_2^2 \\
&\leq \|\widehat{\mathbf{w}}_1 - \mathbf{u}\|_2^2 \qquad\qquad\qquad \left[\text{since } \|\Pi_{\mathcal{W}}(\widehat{\mathbf{w}}_1) - \mathbf{u}\|_2^2 \leq \|\widehat{\mathbf{w}}_1 - \mathbf{u}\|_2^2\right] \\
&\leq \|\mathbf{u}\|_2^2. \qquad\qquad\qquad\qquad\qquad\qquad\qquad\quad \left[\text{since } \widehat{\mathbf{w}}_1 = \mathbf{0}\right]
\end{aligned}
$$

Let $\tilde{\Delta}_t = \|\mathbf{w}_t - \mathbf{u}\|_2^2 - \|\widehat{\mathbf{w}}_{t+1} - \mathbf{u}\|_2^2$, we have

$$\tilde{\Delta}_t \leq \|\mathbf{w}_t - \mathbf{u}\|_2^2 - \|\mathbf{w}_{t+1} - \mathbf{u}\|_2^2 = \Delta_t.$$

Using the recursive updating rule of $\widehat{\mathbf{w}}_{t+1}$ in PA, we rewrite $\tilde{\Delta}_t$ as

$$
\begin{aligned}
\tilde{\Delta}_t &= \|\mathbf{w}_t - \mathbf{u}\|_2^2 - \|\mathbf{w}_t + \text{sign}\left[y_t - \mathbf{w}_t^\mathsf{T}\mathbf{x}_t\right]\xi_t \mathbf{x}_t - \mathbf{u}\|_2^2 \\
&= -\|\text{sign}\left[y_t - \mathbf{w}_t^\mathsf{T}\mathbf{x}_t\right]\xi_t \mathbf{x}_t\|_2^2 - 2(\mathbf{w}_t - \mathbf{u})^\mathsf{T}\text{sign}\left[y_t - \mathbf{w}_t^\mathsf{T}\mathbf{x}_t\right]\xi_t \mathbf{x}_t \\
&= -\xi_t^2 \|\mathbf{x}_t\|_2^2 - 2 \cdot \text{sign}\left[y_t - \mathbf{w}_t^\mathsf{T}\mathbf{x}_t\right]\xi_t \left(\mathbf{w}_t^\mathsf{T}\mathbf{x}_t - \mathbf{u}^\mathsf{T}\mathbf{x}_t\right) \\
&= -\xi_t^2 \|\mathbf{x}_t\|_2^2 - 2 \cdot \text{sign}\left[y_t - \mathbf{w}_t^\mathsf{T}\mathbf{x}_t\right]\xi_t \left(\mathbf{w}_t^\mathsf{T}\mathbf{x}_t - y_t + y_t - \mathbf{u}^\mathsf{T}\mathbf{x}_t\right) \\
&= -\xi_t^2 \|\mathbf{x}_t\|_2^2 - 2 \cdot \text{sign}\left[y_t - \mathbf{w}_t^\mathsf{T}\mathbf{x}_t\right]\xi_t \left(\mathbf{w}_t^\mathsf{T}\mathbf{x}_t - y_t\right) + 2 \cdot \text{sign}\left[y_t - \mathbf{w}_t^\mathsf{T}\mathbf{x}_t\right]\xi_t \left(\mathbf{u}^\mathsf{T}\mathbf{x}_t - y_t\right) \\
&= -\xi_t^2 \|\mathbf{x}_t\|_2^2 + 2\xi_t |\mathbf{w}_t^\mathsf{T}\mathbf{x}_t - y_t| + 2 \cdot \text{sign}\left[y_t - \mathbf{w}_t^\mathsf{T}\mathbf{x}_t\right]\xi_t \left(\mathbf{u}^\mathsf{T}\mathbf{x}_t - y_t\right) \\
&\geq -\xi_t^2 \|\mathbf{x}_t\|_2^2 + 2\xi_t(\ell_t + \varepsilon) - 2\xi_t(\ell_t^\star + \varepsilon) \\
&= \xi_t \left( 2\ell_t - \xi_t \|\mathbf{x}_t\|_2^2 - 2\ell_t^\star \right).
\end{aligned}
$$

Therefore, we have

$$\sum_{t=1}^{T} \xi_t \left( 2\ell_t - \xi_t \|\mathbf{x}_t\|_2^2 - 2\ell_t^\star \right) \leq \sum_{t=1}^{T} \tilde{\Delta}_t \leq \sum_{t=1}^{T} \Delta_t \leq \|\mathbf{u}\|_2^2.$$

$\square$

# E   Robust Analysis on Performance of APAS on Heavy-tailed Data

To demonstrate the performance of APAS in the presence of high volatility and noisy data, we conducted simulations following the same procedure as in our synthetic data experiments outlined in Section 4.1.

In Section 4.1, we considered Gaussian noise $\omega \sim \mathcal{N}(0, \delta^2)$ and Gaussian-distributed side information data $\mathbf{r}_t \sim \mathcal{N}(\boldsymbol{\mu}, \boldsymbol{\Sigma})$. However, heavy-tailed distributions are generally considered more realistic models of data-generating processes in financial markets than Gaussian distributions (Cardoso et al., 2021, 2022). To evaluate the performance of APAS with highly volatile and noisy data, we generate heavy-tailed noise $\omega$ and data $\mathbf{r}_t$ based on the Student's $t$-distribution, using the same mean and variance settings. The degree of freedom for the Student's t-distribution is set to 3, representing significant heavy tails.

Table 2 shows the tracking error of APAS with different combinations of noise and data distributions. Specifically, the column "$\mathcal{N}$ noise + $t$ data" means the noise $\omega$ is generated by Gaussian distribution and the side information data $\mathbf{r}_t$ is generated by Student's $t$-distribution. In general, the difference in tracking error is small, indicating the robust performance of APAS in highly volatile data scenarios.

|  | $\mathcal{N}$ **noise +** $\mathcal{N}$ **data** | $\mathcal{N}$ **noise +** $t$ **data** | $t$ **noise +** $\mathcal{N}$ **data** | $t$ **noise +** $t$ **data** |
|---|---|---|---|---|
| $\lambda = 1 \times 10^{-2}$ | 0.000157 | 0.000159 | 0.000164 | 0.000174 |
| $\lambda = 5 \times 10^{-2}$ | 0.000198 | 0.000206 | 0.000198 | 0.000204 |
| $\lambda = 1 \times 10^{-1}$ | 0.000261 | 0.000271 | 0.000259 | 0.000261 |
| $\lambda = 2 \times 10^{-1}$ | 0.000354 | 0.000369 | 0.000353 | 0.000346 |

Table 2: Tracking error of APAS under different combinations of noise and data distributions.

Table 3 compares the excess cumulative return under different combinations of noise and data distributions. For heavy-tailed noise (i.e., $t$-distribution noise), there is a mild performance degradation. Interestingly, for heavy-tailed data, there is a modest improvement, illustrating the robustness of APAS. This is mainly due to the increased chances of outliers in positive side performance for heavy-tailed data. These results show that APAS is robust to heavy-tailed data with adaptivity in tilting the weight towards positive side information.

|  | $\mathcal{N}$ **noise +** $\mathcal{N}$ **data** | $\mathcal{N}$ **noise +** $t$ **data** | $t$ **noise +** $\mathcal{N}$ **data** | $t$ **noise +** $t$ **data** |
|---|---|---|---|---|
| $\lambda = 1 \times 10^{-2}$ | 0.000157 | 0.000159 | 0.000164 | 0.000174 |
| $\lambda = 5 \times 10^{-2}$ | 0.000198 | 0.000206 | 0.000198 | 0.000204 |
| $\lambda = 1 \times 10^{-1}$ | 0.000261 | 0.000271 | 0.000259 | 0.000261 |
| $\lambda = 2 \times 10^{-1}$ | 0.000354 | 0.000369 | 0.000353 | 0.000346 |

Table 3: Excess cumulative return of APAS under different combinations of noise and data distributions.

